# Max-Margin Markov Networks

**Ben Taskar    Carlos Guestrin    Daphne Koller**
{*btaskar,guestrin,koller*}*@cs.stanford.edu*
Stanford University

## Abstract

In typical classification tasks, we seek a function which assigns a label to a single object. Kernel-based approaches, such as support vector machines (SVMs), which maximize the margin of confidence of the classifier, are the method of choice for many such tasks. Their popularity stems both from the ability to use high-dimensional feature spaces, and from their strong theoretical guarantees. However, many real-world tasks involve sequential, spatial, or structured data, where multiple labels must be assigned. Existing kernel-based methods ignore structure in the problem, assigning labels independently to each object, losing much useful information. Conversely, probabilistic graphical models, such as Markov networks, can represent correlations between labels, by exploiting problem structure, but cannot handle high-dimensional feature spaces, and lack strong theoretical generalization guarantees. In this paper, we present a new framework that combines the advantages of both approaches: *Maximum margin Markov ($M^3$) networks* incorporate both kernels, which efficiently deal with high-dimensional features, and the ability to capture correlations in structured data. We present an efficient algorithm for learning $M^3$ networks based on a compact quadratic program formulation. We provide a new theoretical bound for generalization in structured domains. Experiments on the task of handwritten character recognition and collective hypertext classification demonstrate very significant gains over previous approaches.

## 1   Introduction

In supervised classification, our goal is to classify instances into some set of discrete categories. Recently, support vector machines (SVMs) have demonstrated impressive successes on a broad range of tasks, including document categorization, character recognition, image classification, and many more. SVMs owe a great part of their success to their ability to use kernels, allowing the classifier to exploit a very high-dimensional (possibly even infinite-dimensional) feature space. In addition to their empirical success, SVMs are also appealing due to the existence of strong generalization guarantees, derived from the margin-maximizing properties of the learning algorithm.

However, many supervised learning tasks exhibit much richer structure than a simple categorization of instances into one of a small number of classes. In some cases, we might need to label a set of inter-related instances. For example: optical character recognition (OCR) or part-of-speech tagging both involve labeling an entire sequence of elements into some number of classes; image segmentation involves labeling all of the pixels in an image; and collective webpage classification involves labeling an entire set of interlinked webpages. In other cases, we might want to label an instance (e.g., a news article) with multiple non-exclusive labels. In both of these cases, we need to assign multiple labels simultaneously, leading to a classification problem that has an exponentially large set of joint

labels. A common solution is to treat such problems as a set of independent classification tasks, dealing with each instance in isolation. However, it is well-known that this approach fails to exploit significant amounts of correlation information [7].

An alternative approach is offered by the probabilistic framework, and specifically by probabilistic graphical models. In this case, we can define and learn a joint probabilistic model over the set of label variables. For example, we can learn a hidden Markov model, or a conditional random field (CRF) [7] over the labels and features of a sequence, and then use a probabilistic inference algorithm (such as the Viterbi algorithm) to classify these instances *collectively*, finding the most likely joint assignment to all of the labels simultaneously. This approach has the advantage of exploiting the correlations between the different labels, often resulting in significant improvements in accuracy over approaches that classify instances independently [7, 10]. The use of graphical models also allows problem structure to be exploited very effectively. Unfortunately, even probabilistic graphical models that are trained discriminatively do not usually achieve the same level of generalization accuracy as SVMs, especially when kernel features are used. Moreover, they are not (yet) associated with generalization bounds comparable to those of margin-based classifiers.

Clearly, the frameworks of kernel-based and probabilistic classifiers offer complementary strengths and weaknesses. In this paper, we present *maximum margin Markov (M$^3$) networks*, which unify the two frameworks, and combine the advantages of both. Our approach defines a log-linear Markov network over a set of label variables (e.g., the labels of the letters in an OCR problem); this network allows us to represent the correlations between these label variables. We then define a margin-based optimization problem for the parameters of this model. For Markov networks that can be triangulated tractably, the resulting quadratic program (QP) has an equivalent *polynomial*-size formulation (e.g., linear for sequences) that allows a very effective solution. By contrast, previous margin-based formulations for sequence labeling [3, 1] require an *exponential* number of constraints. For non-triangulated networks, we provide an approximate reformulation based on the relaxation used by belief propagation algorithms [8, 12]. Importantly, the resulting QP supports the same kernel trick as do SVMs, allowing probabilistic graphical models to inherit the important benefits of kernels. We also show a generalization bound for such margin-based classifiers. Unlike previous results [3], our bound grows logarithmically rather than linearly with the number of label variables. Our experimental results on character recognition and on hypertext classification, demonstrate dramatic improvements in accuracy over both kernel-based instance-by-instance classification and probabilistic models.

## 2 Structure in classification problems

In supervised classification, the task is to learn a function $h : \mathcal{X} \mapsto \mathcal{Y}$ from a set of $m$ i.i.d. instances $S = \{(\mathbf{x}^{(i)}, \mathbf{y}^{(i)} = \mathbf{t}(\mathbf{x}^{(i)}))\}_{i=1}^{m}$, drawn from a fixed distribution $D_{\mathcal{X} \times \mathcal{Y}}$. The classification function $h$ is typically selected from some parametric family $\mathcal{H}$. A common choice is the *linear family*: Given $n$ real-valued basis functions $f_j : \mathcal{X} \times \mathcal{Y} \mapsto \mathbb{R}$, a hypothesis $h_\mathbf{w} \in \mathcal{H}$ is defined by a set of $n$ coefficients $w_j$ such that:

$$h_\mathbf{w}(\mathbf{x}) = \arg\max_{\mathbf{y}} \sum_{i=1}^{n} w_j f_j(\mathbf{x}, \mathbf{y}) = \arg\max_{\mathbf{y}} \mathbf{w}^\top \mathbf{f}(\mathbf{x}, \mathbf{y}) , \qquad (1)$$

where the $\mathbf{f}(\mathbf{x}, \mathbf{y})$ are *features* or *basis functions*.

The most common classification setting — single-label classification — takes $\mathcal{Y} = \{y_1, \ldots, y_k\}$. In this paper, we consider the much more general setting of multi-label classification, where $\mathcal{Y} = \mathcal{Y}_1 \times \ldots \times \mathcal{Y}_l$ with $\mathcal{Y}_i = \{y_1, \ldots, y_k\}$. In an OCR task, for example, each $\mathcal{Y}_i$ is a character, while $\mathcal{Y}$ is a full word. In a webpage collective classification task [10], each $\mathcal{Y}_i$ is a webpage label, whereas $\mathcal{Y}$ is a joint label for an entire website. In these cases, the number of possible assignments to $\mathcal{Y}$ is exponential in the number of labels $l$. Thus, both representing the basis functions $f_j(\mathbf{x}, \mathbf{y})$ in (1) and computing the maximization $\arg\max_\mathbf{y}$ are infeasible.

An alternative approach is based on the framework of probabilistic graphical models. In this case, the model defines (directly or indirectly) a conditional distribution $P(\mathcal{Y} \mid \mathcal{X})$. We can then select the label $\arg\max_{\mathbf{y}} P(\mathbf{y} \mid \mathbf{x})$. The advantage of the probabilistic framework is that it can exploit sparseness in the correlations between labels $\mathcal{Y}_i$. For example, in the OCR task, we might use a Markov model, where $\mathcal{Y}_i$ is conditionally independent of the rest of the labels given $\mathcal{Y}_{i-1}, \mathcal{Y}_{i+1}$.

We can encode this structure using a *Markov network*. In this paper, purely for simplicity of presentation, we focus on the case of pairwise interactions between labels. We emphasize that our results extend easily to the general case. A *pairwise Markov network* is defined as a graph $\mathcal{G} = (\mathcal{Y}, E)$, where each edge $(i, j)$ is associated with a potential function $\psi_{ij}(\mathbf{x}, y_i, y_j)$. The network encodes a joint conditional probability distribution as $P(\mathbf{y} \mid \mathbf{x}) \propto \prod_{(i,j)\in E} \psi_{ij}(\mathbf{x}, y_i, y_j)$. These networks exploit the interaction structure to parameterize a classifier very compactly. In many cases (e.g., tree-structured networks), we can use effective dynamic programming algorithms (such as the Viterbi algorithm) to find the highest probability label $\mathbf{y}$; in others, we can use approximate inference algorithms that also exploit the structure [12].

The Markov network distribution is simply a log-linear model, with the pairwise potential $\psi_{ij}(\mathbf{x}, y_i, y_j)$ representing (in log-space) a sum of basis functions over $\mathbf{x}, y_i, y_j$. We can therefore parameterize such a model using a set of pairwise basis functions $f(\mathbf{x}, y_i, y_j)$ for $(i, j) \in E$. We assume for simplicity of notation that all edges in the graph denote the same type of interaction, so that we can define a set of features

$$f_k(\mathbf{x}, \mathbf{y}) = \sum_{(i,j)\in E} f_k(\mathbf{x}, y_i, y_j). \qquad (2)$$

The network potentials are then $\psi_{ij}(\mathbf{x}, y_i, y_j) = \exp\left[\sum_{k=1}^n w_k f_k(\mathbf{x}, y_i, y_j)\right] = \exp\left[\mathbf{w}^\top \mathbf{f}(\mathbf{x}, y_i, y_j)\right]$.

The parameters $\mathbf{w}$ in a log-linear model can be trained to fit the data, typically by maximizing the likelihood or conditional likelihood (e.g., [7, 10]). This paper presents an algorithm for selecting $\mathbf{w}$ that maximize the margin, gaining all of the advantages of SVMs.

## 3   Margin-based structured classification

For a single-label binary classification problem, support vector machines (SVMs) [11] provide an effective method of learning a maximum-margin decision boundary. For single-label multi-class classification, Crammer and Singer [5] provide a natural extension of this framework by maximizing the margin $\gamma$ subject to constraints:

$$\text{maximize} \quad \gamma \quad \text{s.t.} \quad ||\mathbf{w}|| \le 1; \quad \mathbf{w}^\top \Delta\mathbf{f}_{\mathbf{x}}(\mathbf{y}) \ge \gamma, \quad \forall\, \mathbf{x} \in S, \quad \forall \mathbf{y} \ne \mathbf{t}(\mathbf{x}); \qquad (3)$$

where $\Delta\mathbf{f}_{\mathbf{x}}(\mathbf{y}) = \mathbf{f}(\mathbf{x}, \mathbf{t}(\mathbf{x})) - \mathbf{f}(\mathbf{x}, \mathbf{y})$. The constraints in this formulation ensure that $\arg\max_{\mathbf{y}} \mathbf{w}^\top \mathbf{f}(\mathbf{x}, \mathbf{y}) = \mathbf{t}(\mathbf{x})$. Maximizing $\gamma$ magnifies the difference between the value of the true label and the best runner-up, increasing the "confidence" of the classification.

In structured problems, where we are predicting multiple labels, the loss function is usually not simple 0-1 loss $I(\arg\max_{\mathbf{y}} \mathbf{w}^\top \mathbf{f}_{\mathbf{x}}(\mathbf{y}) = \mathbf{t}(\mathbf{x}))$, but per-label loss, such as the proportion of incorrect labels predicted. In order to extend the margin-based framework to the multi-label setting, we must generalize the notion of margin to take into account the number of labels in $\mathbf{y}$ that are misclassified. In particular, we would like the margin between $\mathbf{t}(\mathbf{x})$ and $\mathbf{y}$ to scale linearly with the number of wrong labels in $\mathbf{y}$, $\Delta\mathbf{t}_{\mathbf{x}}(\mathbf{y})$:

$$\text{maximize} \quad \gamma \quad \text{s.t.} \quad ||\mathbf{w}|| \le 1; \quad \mathbf{w}^\top \Delta\mathbf{f}_{\mathbf{x}}(\mathbf{y}) \ge \gamma\, \Delta\mathbf{t}_{\mathbf{x}}(\mathbf{y}), \quad \forall \mathbf{x} \in S, \quad \forall\, \mathbf{y}; \qquad (4)$$

where $\Delta\mathbf{t}_{\mathbf{x}}(\mathbf{y}) = \sum_{i=1}^l \Delta\mathbf{t}_{\mathbf{x}}(y_i)$ and $\Delta\mathbf{t}_{\mathbf{x}}(y_i) \equiv I(y_i \ne (\mathbf{t}(\mathbf{x}))_i)$. Now, using a standard transformation to eliminate $\gamma$, we get a quadratic program (QP):

$$\text{minimize} \quad \frac{1}{2}||\mathbf{w}||^2 \quad \text{s.t.} \quad \mathbf{w}^\top \Delta\mathbf{f}_{\mathbf{x}}(\mathbf{y}) \ge \Delta\mathbf{t}_{\mathbf{x}}(\mathbf{y}), \quad \forall \mathbf{x} \in S, \quad \forall\, \mathbf{y}. \qquad (5)$$

Unfortunately, the data is often not separable by a hyperplane defined over the space of the given set of features. In such cases, we need to introduce slack variables $\xi_{\mathbf{x}}$ to allow

some constraints to be violated. We can now present the complete form of our optimization problem, as well as the equivalent dual problem [2]:

| Primal formulation (6) | Dual formulation (7) |
|---|---|

$$\min \quad \frac{1}{2}||\mathbf{w}||^2 + C\sum_{\mathbf{x}}\xi_{\mathbf{x}} \; ; \qquad \max \quad \sum_{\mathbf{x},\mathbf{y}}\alpha_{\mathbf{x}}(\mathbf{y})\Delta\mathbf{t}_{\mathbf{x}}(\mathbf{y}) - \frac{1}{2}\left\|\left|\sum_{\mathbf{x},\mathbf{y}}\alpha_{\mathbf{x}}(\mathbf{y})\Delta\mathbf{f}_{\mathbf{x}}(\mathbf{y})\right|\right\|^2 ;$$

$$\text{s.t. } \mathbf{w}^\top\Delta\mathbf{f}_{\mathbf{x}}(\mathbf{y}) \geq \Delta\mathbf{t}_{\mathbf{x}}(\mathbf{y}) - \xi_{\mathbf{x}}, \; \forall\mathbf{x},\mathbf{y}. \quad \text{s.t. } \sum_{\mathbf{y}}\alpha_{\mathbf{x}}(\mathbf{y}) = C, \forall\mathbf{x}; \;\; \alpha_{\mathbf{x}}(\mathbf{y}) \geq 0 \,, \forall\mathbf{x},\mathbf{y}.$$

(Note: for each $\mathbf{x}$, we add an extra dual variable $\alpha_{\mathbf{x}}(\mathbf{t}(\mathbf{x}))$, with no effect on the solution.)

## 4 Exploiting structure in M$^3$ networks

Unfortunately, both the number of constraints in the primal QP in (6), and the number of variables in the dual QP in (7) are exponential in the number of labels $l$. In this section, we present an equivalent, polynomially-sized, formulation.

Our main insight is that the variables $\alpha_{\mathbf{x}}(\mathbf{y})$ in the dual formulation (7) can be interpreted as a density function over $\mathbf{y}$ conditional on $\mathbf{x}$, as $\sum_{\mathbf{y}}\alpha_{\mathbf{x}}(\mathbf{y}) = C$ and $\alpha_{\mathbf{x}}(\mathbf{y}) \geq 0$. The dual objective is a function of expectations of $\Delta\mathbf{t}_{\mathbf{x}}(\mathbf{y})$ and $\Delta\mathbf{f}_{\mathbf{x}}(\mathbf{y})$ with respect to $\alpha_{\mathbf{x}}(\mathbf{y})$. Since both $\Delta\mathbf{t}_{\mathbf{x}}(\mathbf{y}) = \sum_i \Delta\mathbf{t}_{\mathbf{x}}(y_i)$ and $\Delta\mathbf{f}_{\mathbf{x}}(\mathbf{y}) = \sum_{(i,j)}\Delta\mathbf{f}_{\mathbf{x}}(y_i, y_j)$ are sums of functions over nodes and edges, we only need node and edge marginals of the measure $\alpha_{\mathbf{x}}(\mathbf{y})$ to compute their expectations. We define the marginal dual variables as follows:

$$\begin{aligned} \mu_{\mathbf{x}}(y_i, y_j) &= \sum_{\mathbf{y}\sim[y_i,y_j]}\alpha_{\mathbf{x}}(\mathbf{y}), & \forall\,(i,j) \in E, \forall y_i, y_j, \;\forall\,\mathbf{x}; \\ \mu_{\mathbf{x}}(y_i) &= \sum_{\mathbf{y}\sim[y_i]}\alpha_{\mathbf{x}}(\mathbf{y}), & \forall\,i, \;\forall y_i, \;\forall\,\mathbf{x}; \end{aligned} \qquad (8)$$

where $\mathbf{y} \sim [y_i, y_j]$ denotes a full assignment $\mathbf{y}$ consistent with partial assignment $y_i, y_j$.

Now we can reformulate our entire QP (7) in terms of these dual variables. Consider, for example, the first term in the objective function:

$$\sum_{\mathbf{y}}\alpha_{\mathbf{x}}(\mathbf{y})\Delta\mathbf{t}_{\mathbf{x}}(\mathbf{y}) = \sum_{\mathbf{y}}\sum_i\alpha_{\mathbf{x}}(\mathbf{y})\Delta\mathbf{t}_{\mathbf{x}}(y_i) = \sum_{i,y_i}\Delta\mathbf{t}_{\mathbf{x}}(y_i)\sum_{\mathbf{y}\sim[y_i]}\alpha_{\mathbf{x}}(\mathbf{y}) = \sum_{i,y_i}\mu_{\mathbf{x}}(y_i)\Delta\mathbf{t}_{\mathbf{x}}(y_i).$$

The decomposition of the second term in the objective uses edge marginals $\mu_{\mathbf{x}}(y_i, y_j)$.

In order to produce an equivalent QP, however, we must also ensure that the dual variables $\mu_{\mathbf{x}}(y_i, y_j), \mu_{\mathbf{x}}(y_i)$ are the marginals resulting from a legal density $\alpha(\mathbf{y})$; that is, that they belong to the *marginal polytope* [4]. In particular, we must enforce consistency between the pairwise and singleton marginals (and hence between overlapping pairwise marginals):

$$\sum_{y_i}\mu_{\mathbf{x}}(y_i, y_j) = \mu_{\mathbf{x}}(y_j), \;\forall y_j, \;\;\forall(i,j) \in E, \;\forall\mathbf{x}. \qquad (9)$$

If the Markov network for our basis functions is a forest (singly connected), these constraints are equivalent to the requirement that the $\mu$ variables arise from a density. Therefore, the following factored dual QP is equivalent to the original dual QP:

$$\max \quad \sum_{\mathbf{x}}\sum_{i,y_i}\mu_{\mathbf{x}}(y_i)\Delta\mathbf{t}_{\mathbf{x}}(y_i) - \frac{1}{2}\sum_{\mathbf{x},\hat{\mathbf{x}}}\sum_{\substack{(i,j)\\y_i,y_j}}\sum_{\substack{(r,s)\\y_r,y_s}}\mu_{\mathbf{x}}(y_i,y_j)\mu_{\hat{\mathbf{x}}}(y_r,y_s)\mathbf{f}_{\mathbf{x}}(y_i,y_j)^\top\mathbf{f}_{\hat{\mathbf{x}}}(y_r,y_s);$$

$$\text{s.t.} \quad \sum_{y_i}\mu_{\mathbf{x}}(y_i,y_j) = \mu_{\mathbf{x}}(y_j); \quad \sum_{y_i}\mu_{\mathbf{x}}(y_i) = C; \quad \mu_{\mathbf{x}}(y_i,y_j) \geq 0. \qquad (10)$$

Similarly, the original primal can be factored as follows:

$$\min \quad \frac{1}{2}||\mathbf{w}||^2 + C\sum_{\mathbf{x}}\sum_i\xi_{\mathbf{x},i} + C\sum_{\mathbf{x}}\sum_{(i,j)}\xi_{\mathbf{x},ij};$$

$$\text{s.t.} \quad \mathbf{w}^\top\Delta\mathbf{f}_{\mathbf{x}}(y_i,y_j) + \sum_{(i',j):i'\neq i}m_{\mathbf{x},i'}(y_j) + \sum_{(j',i):j'\neq j}m_{\mathbf{x},j'}(y_i) \geq -\xi_{\mathbf{x},ij};$$

$$\sum_{(i,j)}m_{\mathbf{x},j}(y_i) \geq \Delta\mathbf{t}_{\mathbf{x}}(y_i) - \xi_{\mathbf{x},i}; \qquad \xi_{\mathbf{x},ij} \geq 0, \xi_{\mathbf{x},i} \geq 0. \qquad (11)$$

The solution to the factored dual gives us: $\mathbf{w} = \sum_{\mathbf{x}} \sum_{(i,j)} \sum_{y_i, y_j} \mu_{\mathbf{x}}(y_i, y_j) \Delta \mathbf{f}_{\mathbf{x}}(y_i, y_j)$.

**Theorem 4.1** *If for each $\mathbf{x}$ the edges $E$ form a forest, then a set of weights $\mathbf{w}$ will be optimal for the QP in (6) if and only if it is optimal for the factored QP in (11).* ∎

If the underlying Markov net is not a forest, then the constraints in (9) are not sufficient to enforce the fact that the $\mu$'s are in the marginal polytope. We can address this problem by triangulating the graph, and introducing new $\eta$ LP variables that now span larger subsets of $Y_i$'s. For example, if our graph is a 4-cycle $Y_1$—$Y_2$—$Y_3$—$Y_4$—$Y_1$, we might triangulate the graph by adding an arc $Y_1$—$Y_3$, and introducing $\eta$ variables over joint instantiations of the cliques $Y_1, Y_2, Y_3$ and $Y_1, Y_3, Y_4$. These new $\eta$ variables are used in linear equalities that constrain the original $\mu$ variables to be consistent with a density. The $\eta$ variables appear only in the constraints; they do not add any new basis functions nor change the objective function. The number of constraints introduced is exponential in the number of variables in the new cliques. Nevertheless, in many classification problems, such as sequences and other graphs with low tree-width [4], the extended QP can be solved efficiently.

Unfortunately, triangulation is not feasible in highly connected problems. However, we can still solve the QP in (10) defined by an untriangulated graph with loops. Such a procedure, which enforces only local consistency of marginals, optimizes our objective only over a relaxation of the marginal polytope. In this way, our approximation is analogous to the approximate belief propagation (BP) algorithm for inference in graphical models [8]. In fact, BP makes an additional approximation, using not only the relaxed marginal polytope but also an approximate objective (Bethe free-energy) [12]. Although the approximate QP does not offer the theoretical guarantee in Theorem 4.1, the solutions are often very accurate in practice, as we demonstrate below.

As with SVMs [11], the factored dual formulation in (10) uses only dot products between basis functions. This allows us to use a kernel to define very large (and even infinite) set of features. In particular, we define our basis functions by $\mathbf{f}_{\mathbf{x}}(y_i, y_j) = \rho(y_i, y_j)\phi^{ij}(\mathbf{x})$, i.e., the product of a selector function $\rho(y_i, y_j)$ with a possibly infinite feature vector $\phi^{ij}(\mathbf{x})$. For example, in the OCR task, $\rho(y_i, y_j)$ could be an indicator function over the class of two adjacent characters $i$ and $j$, and $\phi^{ij}(\mathbf{x})$ could be an RBF kernel on the images of these two characters. The operation $\mathbf{f}_{\mathbf{x}}(y_i, y_j)^{\top} \mathbf{f}_{\hat{\mathbf{x}}}(y_r, y_s)$ used in the objective function of the factored dual QP is now $\rho(y_i, y_j)\rho(y_r, y_s)K_{\phi}(\mathbf{x}, i, j, \hat{\mathbf{x}}, r, s)$, where $K_{\phi}(\mathbf{x}, i, j, \hat{\mathbf{x}}, r, s) = \phi^{ij}(\mathbf{x}) \cdot \phi^{rs}(\mathbf{x})$ is the kernel function for the feature $\phi$. Even for some very complex functions $\phi$, the dot-product required to compute $K_{\phi}$ can be executed efficiently [11].

## 5  SMO learning of M³ networks

Although the number of variables and constraints in the factored dual in (10) is polynomial in the size of the data, the number of coefficients in the quadratic term (kernel matrix) in the objective is quadratic in the number of examples and edges in the network. Unfortunately, this matrix is often too large for standard QP solvers. Instead, we use a coordinate descent method analogous to the sequential minimal optimization (SMO) used for SVMs [9].

Let us begin by considering the original dual problem (7). The SMO approach solves this QP by analytically optimizing two-variable subproblems. Recall that $\sum_{\mathbf{y}} \alpha_{\mathbf{x}}(\mathbf{y}) = C$. We can therefore take any two variables $\alpha_{\mathbf{x}}(\mathbf{y}^1)$, $\alpha_{\mathbf{x}}(\mathbf{y}^2)$ and "move weight" from one to the other, keeping the values of all other variables fixed. More precisely, we optimize for $\alpha'_{\mathbf{x}}(\mathbf{y}^1), \alpha'_{\mathbf{x}}(\mathbf{y}^2)$ such that $\alpha'_{\mathbf{x}}(\mathbf{y}^1) + \alpha'_{\mathbf{x}}(\mathbf{y}^2) = \alpha_{\mathbf{x}}(\mathbf{y}^1) + \alpha_{\mathbf{x}}(\mathbf{y}^2)$.

Clearly, however, we cannot perform this optimization in terms of the original dual, which is exponentially large. Fortunately, we can perform precisely the same optimization in terms of the marginal dual variables. Let $\lambda = \alpha'_{\mathbf{x}}(\mathbf{y}^1) - \alpha_{\mathbf{x}}(\mathbf{y}^1) = \alpha_{\mathbf{x}}(\mathbf{y}^2) - \alpha'_{\mathbf{x}}(\mathbf{y}^2)$. Consider a dual variable $\mu_{\mathbf{x}}(y_i, y_j)$. It is easy to see that a change from $\alpha_{\mathbf{x}}(\mathbf{y}^1), \alpha_{\mathbf{x}}(\mathbf{y}^2)$ to $\alpha'_{\mathbf{x}}(\mathbf{y}^1), \alpha'_{\mathbf{x}}(\mathbf{y}^2)$ has the following effect on $\mu_{\mathbf{x}}(y_i, y_j)$:

$$\mu'_{\mathbf{x}}(y_i, y_j) = \mu_{\mathbf{x}}(y_i, y_j) + \lambda I(y_i = y_i^1, y_j = y_j^1) - \lambda I(y_i = y_i^2, y_j = y_j^2). \qquad (12)$$

We can solve the one-variable quadratic subproblem in $\lambda$ analytically and update the appropriate $\mu$ variables. We use inference in the network to test for optimality of the current solution (the KKT conditions [2]) and use violations from optimality as a heuristic to select the next pair $\mathbf{y}^1, \mathbf{y}^2$. We omit details for lack of space.

## 6  Generalization bound

In this section, we show a generalization bound for the task of multi-label classification that allows us to relate the error rate on the training set to the generalization error. As we shall see, this bound is significantly stronger than previous bounds for this problem.

Our goal in multi-label classification is to maximize the number of correctly classified labels. Thus an appropriate error function is the *average per-label loss* $\mathcal{L}(\mathbf{w}, \mathbf{x}) = \frac{1}{l}\Delta \mathbf{t_x}(\arg\max_{\mathbf{y}} \mathbf{w}^\top \mathbf{f_x}(\mathbf{y}))$. As in other generalization bounds for margin-based classifiers, we relate the generalization error to the margin of the classifier. In Sec. 3, we define the notion of per-label margin, which grows with the number of mistakes between the correct assignment and the best runner-up. We can now define a *$\gamma$-margin per-label loss*:

$$\mathcal{L}^\gamma(\mathbf{w}, \mathbf{x}) = \sup_{\mathbf{z}:\ |\mathbf{z}(\mathbf{y}) - \mathbf{w}^\top \mathbf{f_x}(\mathbf{y})| \leq \gamma \Delta \mathbf{t_x}(\mathbf{y});\ \forall \mathbf{y}} \frac{1}{l}\Delta \mathbf{t_x}(\arg\max_{\mathbf{y}} \mathbf{z}(\mathbf{y})).$$

This loss function measures the worst per-label loss on $\mathbf{x}$ made by any classifier $z$ which is perturbed from $\mathbf{w}^\top \mathbf{f_x}$ by at most a $\gamma$-margin per-label. We can now prove that the generalization accuracy of any classifier is bounded by its expected $\gamma$-margin per-label loss on the training data, plus a term that grows inversely with the margin.Intuitively, the first term corresponds to the "bias", as margin $\gamma$ decreases the complexity of our hypothesis class by considering a $\gamma$-per-label margin ball around $\mathbf{w}^\top \mathbf{f_x}$ and selecting one (the worst) classifier within this ball. As $\gamma$ shrinks, our hypothesis class becomes more complex, and the first term becomes smaller, but at the cost of increasing the second term, which intuitively corresponds to the "variance". Thus, the result provides a bound to the generalization error that trades off the *effective* complexity of the hypothesis space with the training error.

**Theorem 6.1** *If the edge features have bounded 2-norm,* $\max_{(i,j), y_i, y_j} \|\mathbf{f_x}(y_i, y_j)\|_2 \leq R_{edge}$, *then for a family of hyperplanes parameterized by* $\mathbf{w}$, *and any* $\delta > 0$, *there exists a constant* $K$ *such that for any* $\gamma > 0$ *per-label margin, and* $m > 1$ *samples, the per-label loss is bounded by:*

$$E_{\mathbf{x}}\mathcal{L}(\mathbf{w}, \mathbf{x}) \quad \leq \quad E_S \mathcal{L}^\gamma(\mathbf{w}, \mathbf{x}) + \sqrt{\frac{K}{m}\left[\frac{R_{edge}^2 \|\mathbf{w}\|_2^2 q^2}{\gamma^2}\left[\ln m + \ln l + \ln q + \ln k\right] + \ln\frac{1}{\delta}\right]} \quad ;$$

*with probability at least* $1 - \delta$, *where* $q = \max_i |\{(i,j) \in E\}|$ *is the maximum edge degree in the network,* $k$ *is the number of classes in a label, and* $l$ *is the number of labels.* ∎

Unfortunately, we omit the proof due to lack of space. (See a longer version of the paper at `http://cs.stanford.edu/~btaskar/`.) The proof uses a covering number argument analogous to previous results in SVMs [13]. However we propose a novel method for covering structured problems by constructing a cover to the loss function from a cover of the individual edge basis function differences $\Delta \mathbf{f_x}(y_i, y_j)$. This new type of cover is polynomial in the number of edges, yielding significant improvements in the bound.

Specifically, our bound has a logarithmic dependence on the number of labels ($\ln l$) and depends only on the 2-norm of the basis functions per-edge ($R_{edge}$). This is a significant gain over the previous result of Collins [3] which has linear dependence on the number of labels ($l$), and depends on the joint 2-norm of all of the features (which is $\sim lR_{edge}$, unless each sequence is normalized separately, which is often ineffective in practice). Finally, note that if $\frac{l}{m} = \mathcal{O}(1)$ (for example, in OCR, if the number of instances is at least a constant times the length of a word), then our bound is independent of the number of labels $l$. Such a result was, until now, an open problem for margin-based sequence classification [3].

## 7  Experiments

We evaluate our approach on two very different tasks: a sequence model for handwriting recognition and an arbitrary topology Markov network for hypertext classification.

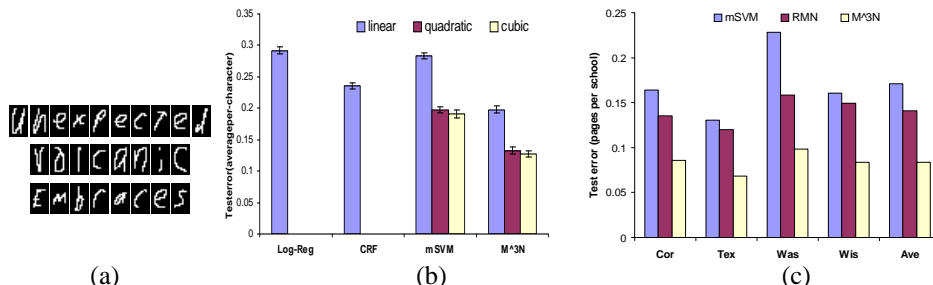

Figure 1: (a) 3 example words from the OCR data set; (b) OCR: Average per-character test error for logistic regression, CRFs, multiclass SVMs, and M³Ns, using linear, quadratic, and cubic kernels; (c) Hypertext: Test error for multiclass SVMs, RMNs and M³Ns, by school and average.

**Handwriting Recognition.** We selected a subset of $\sim 6100$ handwritten words, with average length of $\sim 8$ characters, from 150 human subjects, from the data set collected by Kassel [6]. Each word was divided into characters, each character was rasterized into an image of 16 by 8 binary pixels. (See Fig. 1(a).) In our framework, the image for each word corresponds to $\mathbf{x}$, a label of an individual character to $\mathcal{Y}_i$, and a labeling for a complete word to $\mathcal{Y}$. Each label $\mathcal{Y}_i$ takes values from one of 26 classes $\{a, \ldots, z\}$.

The data set is divided into 10 folds of $\sim 600$ training and $\sim 5500$ testing examples. The accuracy results, summarized in Fig. 1(b), are averages over the 10 folds. We implemented a selection of state-of-the-art classification algorithms: *independent label approaches*, which do not consider the correlation between neighboring characters — logistic regression, multi-class SVMs as described in (3), and one-against-all SVMs (whose performance was slightly lower than multi-class SVMs); and *sequence approaches* — CRFs, and our proposed M³ networks. Logistic regression and CRFs are both trained by maximizing the conditional likelihood of the labels given the features, using a zero-mean diagonal Gaussian prior over the parameters, with a standard deviation between 0.1 and 1. The other methods are trained by margin maximization. Our features for each label $\mathcal{Y}_i$ are the corresponding image of $i$th character. For the sequence approaches (CRFs and M³), we used an indicator basis function to represent the correlation between $\mathcal{Y}_i$ and $\mathcal{Y}_{i+1}$. For margin-based methods (SVMs and M³), we were able to use kernels (both quadratic and cubic were evaluated) to increase the dimensionality of the feature space. Using these high-dimensional feature spaces in CRFs is not feasible because of the enormous number of parameters.

Fig. 1(b) shows two types of gains in accuracy: First, by using kernels, margin-based methods achieve a very significant gain over the respective likelihood maximizing methods. Second, by using sequences, we obtain another significant gain in accuracy. Interestingly, the error rate of our method using linear features is 16% lower than that of CRFs, and about the same as multi-class SVMs with cubic kernels. Once we use cubic kernels our error rate is 45% lower than CRFs and about 33% lower than the best previous approach. For comparison, the previously published results, although using a different setup (e.g., a larger training set), are about comparable to those of multiclass SVMs.

**Hypertext.** We also tested our approach on collective hypertext classification, using the data set in [10], which contains web pages from four different Computer Science departments. Each page is labeled as one of course, faculty, student, project, other. In all of our experiments, we learn a model from three schools, and test on the remaining school. The text content of the web page and anchor text of incoming links is represented using a set of binary attributes that indicate the presence of different words. The baseline model is a simple linear multi-class SVM that uses only words to predict the category of the page. The second model is a *relational* Markov network (RMN) of Taskar *et al.* [10], which in addition to word-label dependence, has an edge with a potential over the labels of two pages that are hyper-linked to each other. This model defines a Markov network over each web site that was trained to maximize the conditional probability of the labels given the words

and the links. The third model is a $M^3$ net with the same features but trained by maximizing the margin using the relaxed dual formulation and loopy BP for inference.

Fig. 1(c) shows a gain in accuracy from SVMs to RMNs by using the correlations between labels of linked web pages, and a very significant additional gain by using maximum margin training. The error rate of M3Ns is $40\%$ lower than that of RMNs, and $51\%$ lower than multi-class SVMs.

## 8 Discussion

We present a discriminative framework for labeling and segmentation of structured data such as sequences, images, etc. Our approach seamlessly integrates state-of-the-art kernel methods developed for classification of independent instances with the rich language of graphical models that can exploit the structure of complex data. In our experiments with the OCR task, for example, our sequence model significantly outperforms other approaches by incorporating high-dimensional decision boundaries of polynomial kernels over character images while capturing correlations between consecutive characters. We construct our models by solving a convex quadratic program that maximizes the *per-label margin*. Although the number of variables and constraints of our QP formulation is polynomial in the example size (e.g., sequence length), we also address its quadratic growth using an effective optimization procedure inspired by SMO. We provide theoretical guarantees on the average *per-label* generalization error of our models in terms of the training set margin. Our generalization bound significantly tightens previous results of Collins [3] and suggests possibilities for analyzing per-label generalization properties of graphical models.

For brevity, we simplified our presentation of graphical models to only pairwise Markov networks. Our formulation and generalization bound easily extend to interaction patterns involving more than two labels (e.g., higher-order Markov models). Overall, we believe that $M^3$ networks will significantly further the applicability of high accuracy margin-based methods to real-world structured data.

**Acknowledgments.** This work was supported by ONR Contract F3060-01-2-0564-P00002 under DARPA's EELD program.

## References

[1] Y. Altun, I. Tsochantaridis, and T. Hofmann. Hidden markov support vector machines. In *Proc. ICML*, 2003.

[2] D. Bertsekas. *Nonlinear Programming*. Athena Scientific, Belmont, MA, 1999.

[3] M. Collins. Parameter estimation for statistical parsing models: Theory and practice of distribution-free methods. In *IWPT*, 2001.

[4] R.G. Cowell, A.P. Dawid, S.L. Lauritzen, and D.J. Spiegelhalter. *Probabilistic Networks and Expert Systems*. Springer, New York, 1999.

[5] K. Crammer and Y. Singer. On the algorithmic implementation of multiclass kernelbased vector machines. *Journal of Machine Learning Research*, 2(5):265–292, 2001.

[6] R. Kassel. *A Comparison of Approaches to On-line Handwritten Character Recognition*. PhD thesis, MIT Spoken Language Systems Group, 1995.

[7] J. Lafferty, A. McCallum, and F. Pereira. Conditional random fields: Probabilistic models for segmenting and labeling sequence data. In *Proc. ICML01*, 2001.

[8] J. Pearl. *Probabilistic Reasoning in Intelligent Systems*. Morgan Kaufmann, 1988.

[9] J. Platt. Using sparseness and analytic QP to speed training of support vector machines. In *NIPS*, 1999.

[10] B. Taskar, P. Abbeel, and D. Koller. Discriminative probabilistic models for relational data. In *Proc. UAI02*, Edmonton, Canada, 2002.

[11] V.N. Vapnik. *The Nature of Statistical Learning Theory*. Springer-Verlag, New York, 1995.

[12] J. Yedidia, W. Freeman, and Y. Weiss. Generalized belief propagation. In *NIPS*, 2000.

[13] T. Zhang. Covering number bounds of certain regularized linear function classes. *Journal of Machine Learning Research*, 2:527–550, 2002.